# Fast Krylov Methods for N-Body Learning

**Nando de Freitas**
Department of Computer Science
University of British Columbia
nando@cs.ubc.ca

**Yang Wang**
School of Computing Science
Simon Fraser University
ywang12@cs.sfu.ca

**Maryam Mahdaviani**
Department of Computer Science
University of British Columbia
maryam@cs.ubc.ca

**Dustin Lang**
Department of Computer Science
University of Toronto
dalang@cs.ubc.ca

## Abstract

This paper addresses the issue of numerical computation in machine learning domains based on similarity metrics, such as kernel methods, spectral techniques and Gaussian processes. It presents a general solution strategy based on Krylov subspace iteration and fast N-body learning methods. The experiments show significant gains in computation and storage on datasets arising in image segmentation, object detection and dimensionality reduction. The paper also presents theoretical bounds on the stability of these methods.

## 1 Introduction

Machine learning techniques based on similarity metrics have gained wide acceptance over the last few years. Spectral clustering [1] is a typical example. Here one forms a Laplacian matrix $\mathbf{L} = \mathbf{D}^{-1/2}\mathbf{W}\mathbf{D}^{-1/2}$, where the entries of $\mathbf{W}$ measure the similarity between data points $\mathbf{x}_i \in \mathcal{X}$, $i = 1, \ldots, N$. For example, a popular choice is to set the entries of $\mathbf{W}$ to

$$w_{ij} = e^{-\frac{1}{\sigma}\|\mathbf{x}_i - \mathbf{x}_j\|^2}$$

where $\sigma$ is a user-specified parameter. $\mathbf{D}$ is a normalizing diagonal matrix with entries $d_i = \sum_j w_{ij}$. The clusters can be found by running, say, K-means on the eigenvectors of $\mathbf{L}$. K-means generates better clusters on this nonlinear embedding of the data provided one adopts a suitable similarity metric.

The list of machine learning domains where one forms a covariance or similarity matrix (be it $\mathbf{W}$, $\mathbf{D}^{-1}\mathbf{W}$ or $\mathbf{D} - \mathbf{W}$) is vast and includes ranking on nonlinear manifolds [2], semi-supervised and active learning [3], Gaussian processes [4], Laplacian eigen-maps [5], stochastic neighbor embedding [6], multi-dimensional scaling, kernels on graphs [7] and many other kernel methods for dimensionality reduction, feature extraction, regression and classification. In these settings, one is interested in either inverting the similarity matrix or finding some of its eigenvectors. The computational cost of both of these operations is $O(N^3)$ while the storage requirement is $O(N^2)$. These costs are prohibitively large in

applications where one encounters massive quantities of data points or where one is interested in real-time solutions such as spectral image segmentation for mobile robots [8]. In this paper, we present general numerical techniques for reducing the computational cost to $O(N \log N)$, or even $O(N)$ in specific cases, and the storage cost to $O(N)$. These reductions are achieved by combining Krylov subspace iterative solvers (such as Arnoldi, Lanczos, GMRES and conjugate gradients) with fast kernel density estimation (KDE) techniques (such as fast multipole expansions, the fast Gauss transform and dual tree recursions [9, 10, 11]).

Specific Krylov methods have been applied to kernel problems. For example, [12] uses Lanczos for spectral clustering and [4] uses conjugate gradients for Gaussian processes. However, the use of fast KDE methods, in particular fast multipole methods, to further accelerate these techniques has only appeared in the context of interpolation [13] and our paper on semi-supervised learning [8]. Here, we go for a more general exposition and present several new examples, such as fast nonlinear embeddings and fast Gaussian processes. More importantly, we attack the issue of stability of these methods. Fast KDE techniques have guaranteed error bounds. However, if these techiques are used inside iterative schemes based on orthogonalization of the Krylov subspace, there is a danger that the errors might grow over iterations. In practice, good behaviour has been observed. In Section 4, we present theoretical results that explain these observations and shed light on the behaviour of these algorithms. Before doing so, we begin with a very brief review of Krylov solvers and fast KDE methods.

## 2  Krylov subspace iteration

This section is a compressed overview of Krylov subspace iteration. The main message is that Krylov methods are very efficient algorithms for solving linear systems and eigenvalue problems, but they require a matrix vector multiplication at each iteration. In the next section, we replace this expensive matrix-vector multiplication with a call to fast KDE routines. Readers happy with this message and familiar with Krylov methods, such as conjugate gradients and Lanczos, can skip the rest of this section.

For ease of presentation, let the similarity matrix be simply $\mathbf{A} = \mathbf{W} \in \mathbb{R}^{N \times N}$, with entries $a_{ij} = a(\mathbf{x}_i, \mathbf{x}_j)$. (One can easily handle other cases, such as $\mathbf{A} = \mathbf{D}^{-1}\mathbf{W}$ and $\mathbf{A} = \mathbf{D} - \mathbf{W}$.) Typical measures of similarity include polynomial $a(\mathbf{x}_i, \mathbf{x}_j) = (\mathbf{x}_i\mathbf{x}_j^T + b)^p$, Gaussian $a(\mathbf{x}_i, \mathbf{x}_j) = e^{-\frac{1}{\sigma}(\mathbf{x}_i - \mathbf{x}_j)(\mathbf{x}_i - \mathbf{x}_j)^T}$ and sigmoid $a(x_i, x_j) = tanh(\alpha\mathbf{x}_i\mathbf{x}_j^T - \beta)$ kernels, where $\mathbf{x}_i\mathbf{x}_j^T$ denotes a scalar inner product. Our goal is to solve linear systems $\mathbf{A}\mathbf{x} = \mathbf{b}$ and (possibly generalized) eigenvalue problems $\mathbf{A}\mathbf{x} = \lambda\mathbf{x}$. The former arise, for example, in semi-supervised learning and Gaussian processes, while the latter arise in spectral clustering and dimensionality reduction. One could attack these problems with naive iterative methods such as the power method, Jacobi and Gauss-Seidel [14]. The problem with these strategies is that the estimate $\mathbf{x}^{(t)}$, at iteration $t$, only depends on the previous estimate $\mathbf{x}^{(t-1)}$. Hence, these methods do typically take too many iterations to converge. It is well accepted in the numerical computation field that Krylov methods [14, 15], which make use of the entire history of solutions $\{\mathbf{x}^{(1)}, \ldots, \mathbf{x}^{(t-1)}\}$, converge at a faster rate.

The intuition behind Krylov subspace methods is to use the history of the solutions we have already computed. We formulate this intuition in terms of projecting an $N$-dimensional problem onto a lower dimensional subspace. Given a matrix $\mathbf{A}$ and a vector $\mathbf{b}$, the associated Krylov matrix is:

$$\mathbf{K} = [\mathbf{b} \quad \mathbf{A}\mathbf{b} \quad \mathbf{A}^2\mathbf{b} \quad \ldots \ ].$$

The *Krylov subspaces* are the spaces spanned by the column vectors of this matrix. In

order to find a new estimate of $\mathbf{x}^{(t)}$ we could project onto the Krylov subspace. However, $\mathbf{K}$ is a poorly conditioned matrix. (As in the power method, $\mathbf{A}^t\mathbf{b}$ is converging to the eigenvector corresponding to the largest eigenvalue of $\mathbf{A}$.) We therefore need to construct a well-conditioned orthogonal matrix $\mathbf{Q}^{(t)} = [\mathbf{q}^{(1)}\cdots\mathbf{q}^{(t)}]$, with $\mathbf{q}^{(i)} \in \mathbb{R}^N$, that spans the Krylov space. That is, the leading $t$ columns of $\mathbf{K}$ and $\mathbf{Q}$ span the same space. This is easily done using the QR-decomposition of $\mathbf{K}$ [14], yielding the following *Arnoldi relation* (augmented Schuur factorization):

$$\mathbf{A}\mathbf{Q}^{(t)} = \mathbf{Q}^{(t+1)}\widetilde{\mathbf{H}}^{(t)},$$

where $\widetilde{\mathbf{H}}^{(t)}$ is the augmented Hessenberg matrix:

$$\widetilde{\mathbf{H}}^{(t)} = \begin{pmatrix} h_{1,1} & h_{1,2} & h_{1,3} & \cdots & h1,t \\ h_{2,1} & h_{2,2} & h_{2,3} & \cdots & h_{2,t} \\ \vdots & \vdots & \vdots & \vdots & \vdots \\ 0 & \cdots & 0 & h_{t,t-1} & h_{t,t} \\ 0 & \cdots & 0 & 0 & h_{t+1,t} \end{pmatrix}.$$

The eigenvalues of the smaller $(t+1) \times t$ Hessenberg matrix approximate the eigenvalues of $\mathbf{A}$ as $t$ increases. These eigenvalues can be computed efficiently by applying the Arnoldi relation recursively as shown in Figure 1. (If $\mathbf{A}$ is symmetric, then $\widetilde{\mathbf{H}}$ is tridiagonal and we obtain the Lanczos algorithm.) *Notice that the matrix vector multiplication $\mathbf{v} = \mathbf{A}\mathbf{q}$ is the expensive step in the Arnoldi algorithm.* Most Krylov algorithms resemble the Arnoldi algorithm in this. To solve systems of equations, we can minimize either the residual

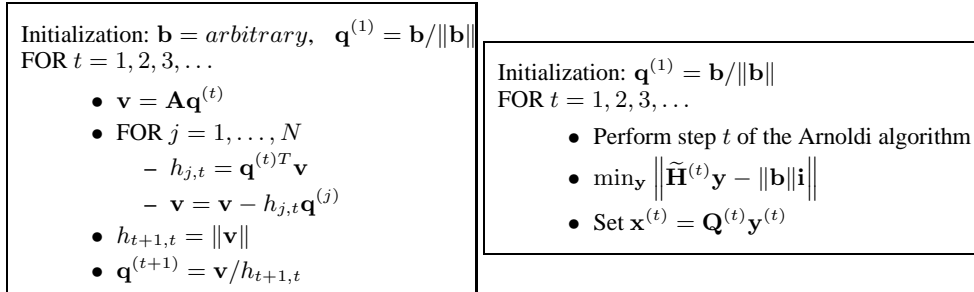

Figure 1: The Arnoldi (left) and GMRES (right) algorithms.

$\mathbf{r}^{(t)} \triangleq \mathbf{b} - \mathbf{A}\mathbf{x}^{(t)}$, leading to the GMRES and MINRES algorithms, or the A-norm, leading to conjugate gradients (CG) [14]. GMRES, MINRES and CG apply to general, symmetric, and spd matrices respectively. For ease of presentation, we focus on the GMRES algorithm.

At step $t$ of GMRES, we approximate the solution by the vector in the Krylov subspace $\mathbf{x}^{(t)} \in \mathcal{K}^{(t)}$ that minimizes the norm of the residual. Since $\mathbf{x}^{(t)}$ is in the Krylov subspace, it can be written as a linear combination of the columns of the Krylov matrix $\mathbf{K}^{(t)}$. Our problem therefore reduces to finding the vector $\mathbf{y} \in \mathbb{R}^t$ that minimizes $\|\mathbf{A}\mathbf{K}^{(t)}\mathbf{y} - \mathbf{b}\|$. As before, stability considerations force us to use the QR decomposition of $\mathbf{K}^{(t)}$. That is, instead of using a linear combination of the columns of $\mathbf{K}^{(t)}$, we use a linear combination of the columns of $\mathbf{Q}^{(t)}$. So our least squares problem becomes $\mathbf{y}^{(t)} = \min_{\mathbf{y}} \|\mathbf{A}\mathbf{Q}^{(t)}\mathbf{y} - \mathbf{b}\|$. Since $\mathbf{A}\mathbf{Q}^{(t)} = \mathbf{Q}^{(t+1)}\widetilde{\mathbf{H}}^{(t)}$, we only need to solve a problem of dimension $(t+1) \times t$: $\mathbf{y}^{(t)} = \min_{\mathbf{y}} \|\mathbf{Q}^{(t+1)}\widehat{\mathbf{H}}^{(t)}\mathbf{y} - \mathbf{b}\|$. Keeping in mind that the columns of the projection matrix $\mathbf{Q}$ are orthonormal, we can rewrite this least squares problem as $\min_{\mathbf{y}} \|\widetilde{\mathbf{H}}^{(t)}\mathbf{y} - \mathbf{Q}^{(t+1)T}\mathbf{b}\|$. We start the iterations with $\mathbf{q}^{(1)} = \mathbf{b}/\|\mathbf{b}\|$ and hence $\mathbf{Q}^{(t+1)T}\mathbf{b} = \|\mathbf{b}\|\mathbf{i}$,

where $\mathbf{i}$ is the unit vector with a 1 in the first entry. The final form of our least squares problem at iteration $t$ is:

$$\mathbf{y}^{(t)} = \min_{\mathbf{y}} \left\| \widetilde{\mathbf{H}}^{(t)}\mathbf{y} - \|\mathbf{b}\|\mathbf{i} \right\|,$$

with solution $\mathbf{x}^{(t)} = \mathbf{Q}^{(t)}\mathbf{y}^{(t)}$. The algorithm is shown in Figure 1. The least squares problem of size $(t+1) \times t$ to compute $\mathbf{y}^{(t)}$ can be solved in $O(t)$ steps using Givens rotations [14]. *Notice again that the expensive step in each iteration is the matrix-vector product $\mathbf{v} = \mathbf{Aq}$. This is true also of CG and other Krylov methods.*

One important property of the Arnoldi relation is that the residuals are orthogonal to the space spanned by the columns of $\mathbf{V} = \mathbf{Q}^{(t+1)}\widetilde{\mathbf{H}}^{(t)}$. That is,

$$\mathbf{V}^T\mathbf{r}^{(t)} = \widetilde{\mathbf{H}}^{(t)T}\mathbf{Q}^{(t+1)T}(\mathbf{b} - \mathbf{Q}^{(t+1)}\widetilde{\mathbf{H}}^{(t)}\mathbf{y}^{(t)}) = \widetilde{\mathbf{H}}^{(t)T}\|\mathbf{b}\|\mathbf{i} - \widetilde{\mathbf{H}}^{(t)T}\widetilde{\mathbf{H}}^{(t)}\mathbf{y}^{(t)} = 0$$

In the following section, we introduce methods to speed up the matrix-vector product $\mathbf{v} = \mathbf{Aq}$. These methods will incur, at most, a pre-specified (tolerance) error $\mathbf{e}^{(t)}$ at iteration $t$. Later, we present theoretical bounds on how these errors affect the residuals and the orthogonality of the Krylov subspace.

## 3  Fast KDE

The expensive step in Krylov methods is the operation $\mathbf{v} = \mathbf{Aq}^{(t)}$. This step requires that we solve two $O(N^2)$ kernel estimates:

$$v_i = \sum_{j=1}^{N} q_j^{(t)} a(\mathbf{x}_i, \mathbf{x}_j) \qquad i = 1, 2, \ldots, M.$$

It is possible to reduce the storage and computational cost to $O(N)$ at the expense of a small specified error tolerance $\epsilon$, say $10^{-6}$, using the *fast Gauss transform* (FGT) algorithm [16, 17]. This algorithm is an instance of more general fast multipole methods for solving $N$-body interactions [9]. The FGT applies when the problem is low dimensional, say $\mathbf{x}_k \in \mathbb{R}^3$. However, to attack larger dimensions one can adopt clustering-based partitions as in the improved fast Gauss transform (IFGT) [10].

Fast multipole methods tend to work only in low dimensions and are specific to the choice of similarity metric. Dual tree recursions based on KD-trees and ball trees [11, 18] overcome these difficulties, but on average cost $O(N \log N)$. Due to space constraints, we can only mention these techniques here, but refer the reader to [18] for a thorough comparison.

## 4  Stability results

The problem with replacing the matrix-vector multiplication at each iteration of the Krylov methods is that we do not know how the errors accumulate over successive iterations. In this section, we will derive bounds that describe what factors influence these errors. In particular, the bounds will state what properties of the similarity metric and measurable quantities affect the residuals and the orthogonality of the Krylov subspaces.

Several papers have addressed the issue of Krylov subspace stability [19, 20, 21]. Our approach follows from [21]. For presentation purposes, we focus on the GMRES algorithm.

Let $\mathbf{e}^{(t)}$ denote the errors introduced in the approximate matrix-vector multiplication at each iteration of Arnoldi. For the purposes of upper-bounding, this is the tolerance of the fast KDE methods. Then, the fast KDE methods change the *Arnoldi relation* to:

$$\mathbf{AQ}^{(t)} + \mathbf{E}^{(t)} = \left[ \mathbf{Aq}^{(1)} + \mathbf{e}^{(1)}, \ldots, \mathbf{Aq}^{(t)} + \mathbf{e}^{(t)} \right] = \mathbf{Q}^{(t+1)}\widetilde{\mathbf{H}}^{(t)},$$

where $\mathbf{E}^{(t)} = \left[\mathbf{e}^{(1)}, \ldots, \mathbf{e}^{(t)}\right]$. The new true residuals are therefore:

$$\mathbf{r}^{(t)} = \mathbf{b} - \mathbf{A}\mathbf{x}^{(t)} = \mathbf{b} - \mathbf{A}\mathbf{Q}^{(t)}\mathbf{y}^{(t)} = \mathbf{b} - \mathbf{Q}^{(t+1)}\widetilde{\mathbf{H}}^{(t)}\mathbf{y}^{(t)} + \mathbf{E}^{(t)}\mathbf{y}^{(t)}$$

and $\widetilde{\mathbf{r}}^{(t)} = \mathbf{b} - \mathbf{Q}^{(t+1)}\widetilde{\mathbf{H}}^{(t)}\mathbf{y}^{(t)}$ are the *measured residuals*.

We need to ensure two bounds when using fast KDE methods in Krylov iterations. First, the measured residuals $\widetilde{\mathbf{r}}^{(t)}$ should not deviate too far from the true residuals $\mathbf{r}^{(t)}$. Second, deviations from orthogonality should be upper-bounded. Let us address the first question. The deviation in residuals is given by

$$\|\widetilde{\mathbf{r}}^{(t)} - \mathbf{r}^{(t)}\| = \|\mathbf{E}^{(t)}\mathbf{y}^{(t)}\|.$$

Let $\mathbf{y}^{(t)} = [y_1, \ldots, y_t]^T$. Then, this deviation satisfies:

$$\|\widetilde{\mathbf{r}}^{(t)} - \mathbf{r}^{(t)}\| = \left\|\sum_{k=1}^{t} y_k \mathbf{e}^{(k)}\right\| \leq \sum_{k=1}^{t} |y_k| \|\mathbf{e}^{(k)}\|. \tag{1}$$

The deviation from orthogonality can be upper-bounded in a similar fashion:

$$\|\mathbf{V}^T\mathbf{r}^{(t)}\| = \|\widetilde{\mathbf{H}}^{(t)T}\mathbf{Q}^{(t+1)T}(\widetilde{\mathbf{r}}^{(t)} + \mathbf{E}^{(t)}\mathbf{y}^{(t)})\| = \left\|\widetilde{\mathbf{H}}^{(t)T}\mathbf{E}^{(t)}\mathbf{y}^{(t)}\right\| \leq \|\widetilde{\mathbf{H}}^{(t)}\| \sum_{k=1}^{t} |y_k| \|\mathbf{e}^{(k)}\| \tag{2}$$

The following lemma provides a relation between the $y_k$ and the measured residuals $\widetilde{\mathbf{r}}^{(k-1)}$.

**Lemma 1.** *[21, Lemma 5.1] Assume that $t$ iterations of the inexact Arnoldi method have been carried out. Then, for any $k = 1, \ldots, t$,*

$$|y_k| \leq \frac{1}{\sigma_t(\widetilde{\mathbf{H}}^{(t)})} \|\widetilde{\mathbf{r}}^{(k-1)}\| \tag{3}$$

*where $\sigma_t(\widetilde{\mathbf{H}}^{(t)})$ denotes the $t$-th singular value of $\widetilde{\mathbf{H}}^{(t)}$.*

The proof of the lemma follows from the QR decomposition of $\widetilde{\mathbf{H}}^{(t)}$, see [15, 21]. This lemma, in conjunction with equations (1) and (2), allows us to establish the main theoretical result of this section:

**Proposition 1.** *Let $\epsilon > 0$. If for every $k \leq t$ we have*

$$\|\mathbf{e}^{(k)}\| < \frac{\sigma_t(\widetilde{\mathbf{H}}^{(t)})}{t} \frac{1}{\|\widetilde{\mathbf{r}}^{(k-1)}\|} \epsilon,$$

*then $\|\widetilde{\mathbf{r}}^{(t)} - \mathbf{r}^{(t)}\| < \epsilon$. Moreover, if*

$$\|\mathbf{e}^{(k)}\| < \frac{\sigma_t(\widetilde{\mathbf{H}}^{(t)})}{t\|\widetilde{\mathbf{H}}^{(t)}\|} \frac{1}{\|\widetilde{\mathbf{r}}^{(k-1)}\|} \epsilon,$$

*then $\|\mathbf{V}^T\mathbf{r}^{(t)}\| < \epsilon$.*

**Proof:** First, we have

$$\|\widetilde{\mathbf{r}}^{(t)} - \mathbf{r}^{(t)}\| \leq \sum_{k=1}^{t} |y_k| \|\mathbf{e}^{(k)}\| < \sum_{k=1}^{t} \frac{\sigma_t(\widetilde{\mathbf{H}}^{(t)})}{t} \frac{1}{\|\widetilde{\mathbf{r}}^{(k-1)}\|} \epsilon \frac{1}{\sigma_t(\widetilde{\mathbf{H}}^{(t)})} \|\widetilde{\mathbf{r}}^{(k-1)}\| = \epsilon.$$

and similarly, $\|\mathbf{V}^T\mathbf{r}^{(t)}\| \leq \|\widetilde{\mathbf{H}}^{(t)}\| \sum_{k=1}^{t} |y_k| \|\mathbf{e}^{(k)}\| < \epsilon$ $\quad\square$

Proposition 1 tells us that in order to keep the residuals bounded while ensuring bounded deviations from orthogonality at iteration $k$, we need to monitor the eigenvalues of $\widetilde{\mathbf{H}}^{(t)}$ and the measured residuals $\widetilde{\mathbf{r}}^{(k-1)}$. Of course, we have no access to $\widetilde{\mathbf{H}}^{(t)}$. However, monitoring the residuals is of practical value. If the residuals decrease, we can increase the tolerance of the fast KDE algorithms and viceversa. The bounds do lead to a natural way of constructing adaptive algorithms for setting the tolerance of the fast KDE algorithms.

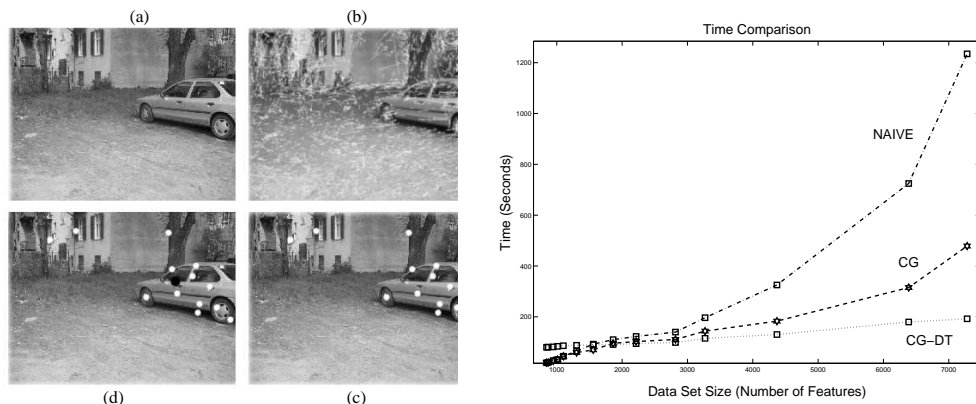

Figure 2: Figure (a) shows a test image from the PASCAL database. Figure (b) shows the SIFT features extracted from the image. Figure (c) shows the positive feature predictions for the label "car". Figure (d) shows the centroid of the positive features as a black dot. The plot on the right shows the computational gains obtained by using fast Krylov methods.

# 5 Experimental results

The results of this section demonstrate that significant computational gains may be obtained by combining fast KDE methods with Krylov iterations. We present results in three domains: spectral clustering and image segmentation [1, 12], Gaussian process regression [4] and stochastic neighbor embedding [6].

## 5.1 Gaussian processes with large dimensional features

In this experiment we use Gaussian processes to predict the labels of 128-dimensional SIFT features [22] for the purposes of object detection and localization as shown in Figure 2. There are typically thousands of features per image, so it is of paramount importance to generate fast predictions. The hard computational task here involves inverting the covariance matrix of the Gaussian process. The figure shows that it is possible to do this efficiently, under the same ROC error, by combining conjugate gradients [4] with dual trees.

## 5.2 Spectral clustering and image segmentation

We applied spectral clustering to color image segmentation; a generalized eigenvalue problem. The types of segmentations obtained are shown in Figure 3. There are no perceptible differences between them. We observed that fast Krylov methods run approximately twice as fast as the Nystrom method. One should note that the result of Nystrom depends on the quality of sampling, while fast N-body methods enable us to work directly with the full matrix, so the solution is less sensitive. Once again, fast KDE methods lead to significant computational improvements over Krylov algorithms (Lanczos in this case).

## 5.3 Stochastic neighbor embedding

Our final example is again a generalized eigenvalue problem arising in dimensionality reduction. We use the stochastic neighbor embedding algorithm of [6] to project two 3-D structures to 2-D, as shown in Figure 4. Again, we observe significant computational improvements.

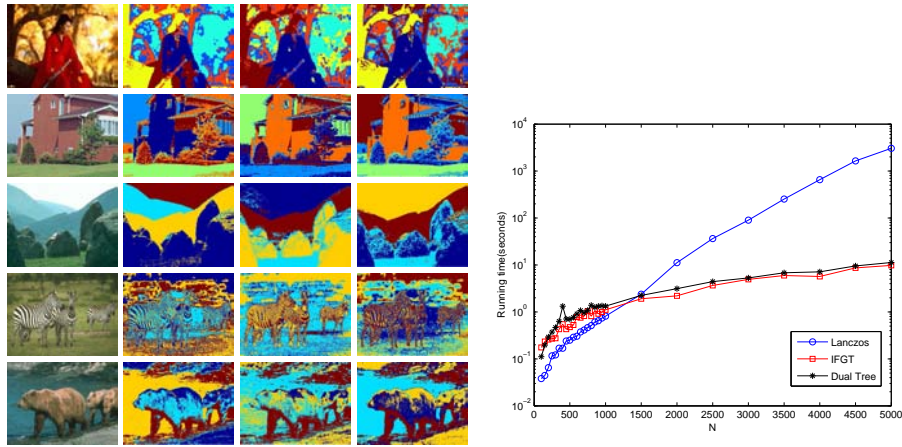

Figure 3: (left) Segmentation results (order: original image, IFGT, dual trees and Nystrom) and (right) computational improvements obtained in spectral clustering.

## 6 Conclusions

We presented a general approach for combining Krylov solvers and fast KDE methods to accelerate machine learning techniques based on similarity metrics. We demonstrated some of the methods on several datasets and presented results that shed light on the stability and convergence properties of these methods. One important point to make is that these methods work better when there is structure in the data. There is no computational gain if there is not statistical information in the data. This is a fascinating relation between computation and statistical information, which we believe deserves further research and understanding. One question is how can we design pre-conditioners in order to improve the convergence behavior of these algorithms. Another important avenue for further research is the application of the bounds presented in this paper in the design of adaptive algorithms.

### Acknowledgments

We would like to thank Arnaud Doucet, Firas Hamze, Greg Mori and Changjiang Yang.

## References

[1] A Y Ng, M I Jordan, and Y Weiss. On spectral clustering: Analysis and algorithm. In *Advances in Neural Information Processing Systems*, pages 849–856, 2001.

[2] D Zhou, J Weston, A Gretton, O Bousquet, and B Scholkopf. Ranking on data manifolds. In *Advances on Neural Information Processing Systems*, 2004.

[3] X Zhu, J Lafferty, and Z Ghahramani. Semi-supervised learning using Gaussian fields and harmonic functions. In *International Conference on Machine Learning*, pages 912–919, 2003.

[4] M N Gibbs. Bayesian Gaussian processes for regression and classification. In *PhD Thesis, University of Cambridge*, 1997.

[5] M Belkin and P Niyogi. Laplacian eigenmaps for dimensionality reduction and data representation. *Neural Computation*, 15(6):1373–1396, 2003.

[6] G Hinton and S Roweis. Stochastic neighbor embedding. In *Advances in Neural Information Processing Systems*, pages 833–840, 2002.

[7] A Smola and R Kondor. Kernels and regularization of graphs. In *Computational Learning Theory*, pages 144–158, 2003.

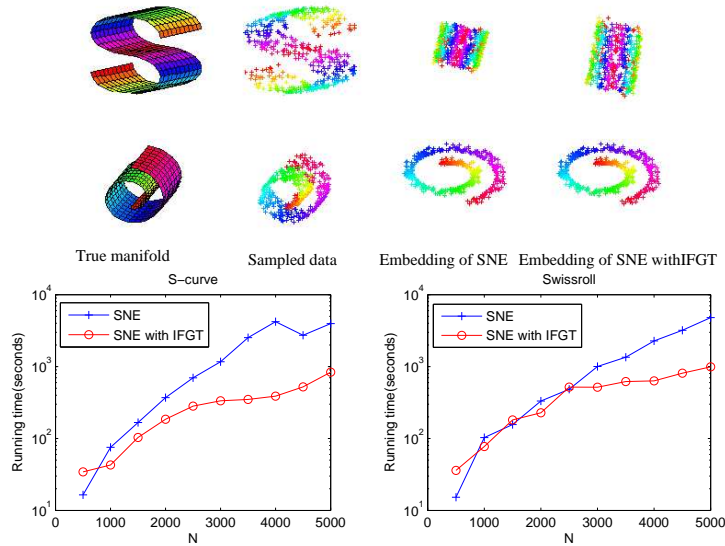

Figure 4: Examples of embedding on S-curve and Swiss-roll datasets.

[8] M Mahdaviani, N de Freitas, B Fraser, and F Hamze. Fast computational methods for visually guided robots. In *IEEE International Conference on Robotics and Automation*, 2004.

[9] L Greengard and V Rokhlin. A fast algorithm for particle simulations. *Journal of Computational Physics*, 73:325–348, 1987.

[10] C Yang, R Duraiswami, N A Gumerov, and L S Davis. Improved fast Gauss transform and efficient kernel density estimation. In *International Conference on Computer Vision*, Nice, 2003.

[11] A Gray and A Moore. Rapid evaluation of multiple density models. In *Artificial Iintelligence and Statistics*, 2003.

[12] J Shi and J Malik. Normalized cuts and image segmentation. In *IEEE Conference on Computer Vision and Pattern Recognition*, pages 731–737, 1997.

[13] R K Beatson, J B Cherrie, and C T Mouat. Fast fitting of radial basis functions: Methods based on preconditioned GMRES iteration. *Advances in Computational Mathematics*, 11:253–270, 1999.

[14] J W Demmel. *Applied Numerical Linear Algebra*. SIAM, 1997.

[15] Y Saad. *Iterative Methods for Sparse Linear Systems*. The PWS Publishing Company, 1996.

[16] L Greengard and J Strain. The fast Gauss transform. *SIAM Journal of Scientific Statistical Computing*, 12(1):79–94, 1991.

[17] B J C Baxter and G Roussos. A new error estimate of the fast Gauss transform. *SIAM Journal of Scientific Computing*, 24(1):257–259, 2002.

[18] D Lang, M Klaas, and N de Freitas. Empirical testing of fast kernel density estimation algorithms. Technical Report TR-2005-03, Department of Computer Science, UBC, 2005.

[19] G H Golub and Q Ye. Inexact preconditioned conjugate gradient method with inner-outer iteration. *SIAM Journal of Scientific Computing*, 21:1305–1320, 1999.

[20] G W Stewart. Backward error bounds for approximate Krylov subspaces. *Linear Algebra and Applications*, 340:81–86, 2002.

[21] V Simoncini and D B Szyld. Theory of inexact Krylov subspace methods and applications to scientific computing. *SIAM Journal on Scientific Computing*, 25:454–477, 2003.

[22] D G Lowe. Object recognition from local scale-invariant features. In *ICCV*, 1999.
